# PAC-Bayesian Model Selection
# for Reinforcement Learning

**Mahdi Milani Fard**
School of Computer Science
McGill University
Montreal, Canada
mmilan1@cs.mcgill.ca

**Joelle Pineau**
School of Computer Science
McGill University
Montreal, Canada
jpineau@cs.mcgill.ca

## Abstract

This paper introduces the first set of PAC-Bayesian bounds for the batch reinforcement learning problem in finite state spaces. These bounds hold regardless of the correctness of the prior distribution. We demonstrate how such bounds can be used for model-selection in control problems where prior information is available either on the dynamics of the environment, or on the value of actions. Our empirical results confirm that PAC-Bayesian model-selection is able to leverage prior distributions when they are informative and, unlike standard Bayesian RL approaches, ignores them when they are misleading.

## 1 Introduction

Bayesian methods in machine learning, although elegant and concrete, have often been criticized not only for their computational cost, but also for their strong assumptions on the correctness of the prior distribution. There are usually no theoretical guarantees when performing Bayesian inference with priors that do not admit the correct posterior. Probably Approximately Correct (PAC) learning techniques, on the other hand, provide distribution-free convergence guarantees with polynomially-bounded sample sizes [1]. These bounds, however, are notoriously loose and impractical. One can argue that such loose bounds are to be expected, as they reflect the inherent difficulty of the problem when no assumptions are made on the distribution of the data.

Both PAC and Bayesian methods have been proposed for reinforcement learning (RL) [2, 3, 4, 5, 6, 7, 8], where an agent is learning to interact with an environment to maximize some objective function. Many of these methods aim to solve the so-called exploration–exploitation problem by balancing the amount of time spent on gathering information about the dynamics of the environment and the time spent acting optimally according to the currently built model. PAC methods are much more conservative than Bayesian methods as they tend to spend more time exploring the system and collecting information [9]. Bayesian methods, on the other hand, are greedier and only solve the problem over a limited planning horizon. As a result of this greediness, Bayesian methods can converge to suboptimal solutions. It has been shown that Bayesian RL is not PAC [9]. We argue here that a more adaptive method can be PAC and at the same time more data efficient if an informative prior is taken into account. Such adaptive techniques have been studied within the PAC-Bayesian literature for supervised learning.

The PAC-Bayesian approach, first introduced by McAllester [10] (extending the work of Shawe-Taylor et al. [11]), combines the distribution-free correctness of PAC theorems with the data-efficiency of Bayesian inference. This is achieved by removing the assumption of the correctness of the prior and, instead, measuring the consistency of the prior over the training data. The empirical results of model selection algorithms for classification tasks using these bounds are comparable to some of the most popular learning algorithms such as AdaBoost and Support Vector Machines [12]. PAC-Bayesian bounds have also been linked to margins in classification tasks [13].

This paper introduces the first results of the application of PAC-Bayesian techniques to the batch RL problem. We derive two PAC-Bayesian bounds on the approximation error in the value function of stochastic policies for reinforcement learning on observable and discrete state spaces. One is a bound on model-based RL where a prior distribution is given on the space of possible models. The second one is for the case of model-free RL, where a prior is given on the space of value functions. In both cases, the bound depends both on an empirical estimate and a measure of distance between the stochastic policy and the one imposed by the prior distribution. We present empirical results where model-selection is performed based on these bounds, and show that PAC-Bayesian bounds follow Bayesian policies when the prior is informative and mimic the PAC policies when the prior is not consistent with the data. This allows us to adaptively balance between the distribution-free correctness of PAC and the data-efficiency of Bayesian inference.

## 2 Background and Notation

In this section, we introduce the notations and definitions used in the paper.

A **Markov Decision Process** (MDP) $M = (S, A, T, R)$ is defined by a set of states $S$, a set of actions $A$, a transition function $T(s, a, s')$ defined as:

$$T(s, a, s') = p(s_{t+1} = s'|s_t = s, a_t = a), \forall s, s' \in S, a \in A, \tag{1}$$

and a (possibly stochastic) reward function $R(s, a) : S \times A \to [R_{\min}, R_{\max}]$. Throughout the paper we assume finite-state, finite-action, discounted-reward MDPs, with the discount factor denoted by $\gamma$. A reinforcement learning agent chooses an action and receives a reward. The environment will then change to a new state according to the transition probabilities.

A **policy** is a (possibly stochastic) function from states to actions. The **value of a state–action pair** $(s, a)$ for policy $\pi$, denoted by $Q^\pi(s, a)$, is the expected discounted sum of rewards ($\sum_t \gamma^t r_t$) if the agent acts according to that policy after taking action $a$ in the first step. The value function satisfies the Bellman equation [14]:

$$Q^\pi(s, a) = R(s, a) + \gamma \sum_{s' \in S} \left(T(s, a, s')Q^\pi(s', \pi(s'))\right). \tag{2}$$

The **optimal policy** is the policy that maximizes the value function. The **optimal value of a state–action pair**, denoted by $Q^*(s, a)$, satisfies the Bellman optimality equation [14]:

$$Q^*(s, a) = R(s, a) + \gamma \sum_{s' \in S} \left(T(s, a, s') \max_{a' \in A} Q^*(s', a')\right). \tag{3}$$

There are many methods developed to find the optimal policy for a given MDP when transition and reward functions are known. Value iteration [14] is a simple dynamic programming method in which one iteratively applies the **Bellman optimality operator**, denoted by $B$, to an initial guess of the optimal value function:

$$BQ(s, a) = R(s, a) + \gamma \sum_{s' \in S} \left(T(s, a, s') \max_{a' \in A} Q(s', a')\right). \tag{4}$$

For simplicity we write $BQ$ when $B$ is applied to the value of all state–action pairs. Since $B$ is a contraction with respect to the infinity norm [15] (i.e. $\|BQ - BQ'\|_\infty \leq \gamma \|Q - Q'\|_\infty$), the value iteration algorithm will converge to the fixed point of the Bellman optimality operator, which is the optimal value function ($BQ^* = Q^*$).

## 3 Model-Based PAC-Bayesian Bound

In model-based RL, one aims to estimate the transition and reward functions and then act optimally according to the estimated models. PAC methods use the empirical average for their estimated model along with frequentist bounds. Bayesian methods use the Bayesian posterior to estimate the model. This section provides a bound that suggests an adaptive method to choose a stochastic estimate between these two extremes, which is both data-efficient and has guaranteed performance.

Assuming that the reward model is known (we make this assumption throughout this section), one can build empirical models of the transition dynamics by gathering sample transitions, denoted by $U$, and taking the empirical average. Let this empirical average model be $\hat{T}(s, a, s') = n_{s,a,s'}/n_{s,a}$, where $n_{s,a,s'}$ and $n_{s,a}$ are the number of corresponding transitions and samples. Trivially, $\mathbb{E}\hat{T} = T$. The empirical value function, denoted by $\hat{Q}$, is defined to be the value function on an MDP with the empirical transition model. As one observes more and more sample trajectories on the MDP, the empirical model gets increasingly more accurate, and so will the empirical value function. Different forms of the following lemma, connecting the error rates on $\hat{T}$ and $\hat{Q}$, are used in many of the PAC-MDP results [4]:

**Lemma 1.** *There is a constant $k \geq (1 - \gamma)^2/\gamma$ such that:*

$$\forall s, a : \|\hat{T}(s, a, .) - T(s, a, .)\|_1 \leq k\epsilon \qquad \Rightarrow \qquad \forall \pi : \|\hat{Q}^\pi - Q^\pi\|_\infty \leq \epsilon. \tag{5}$$

As a consequence of the above lemma, one can act near-optimally in the part of the MDP for which we have gathered enough samples to have a good empirical estimate of the transition model. PAC-MDP methods explicitly [2] or implicitly [3] use that fact to exploit the knowledge on the model as long as they are in the "known" part of the state space. The downside of these methods is that without further assumptions on the model, it will take a large number of sample transitions to get a good empirical estimate of the transition model.

The Bayesian approach to modeling the transition dynamics, on the other hand, starts with a prior distribution over the transition probability and then marginalizes this prior over the data to get a posterior distribution. This is usually done by assuming independent Dirichlet distributions over the transition probabilities, with some initial count vector $\alpha$, and then adding up the observed counts to this initial vector to get the conjugate posterior [6]. The initial $\alpha$-vector encodes the prior knowledge on the transition probabilities, and larger initial values further bias the empirical observation towards the initial belief.

If a strong prior is close to the true values, the Bayesian posterior will be more accurate than the empirical point estimate. However, a strong prior peaked on the wrong values will bias the Bayesian model away from the correct probabilities. Therefore, the Bayesian posterior might not provide the optimal estimate of the model parameters. A good posterior distribution might be somewhere between the empirical point estimate and the Bayesian posterior.

The following theorem is the first PAC-Bayesian bound on the estimation error in the value function when we build a stochastic policy[1] based on some arbitrary posterior distribution $M_q$.

**Theorem 2.** *Let $\pi^*_{T'}$ be the optimal policy with respect to the MDP with transition model $T'$ and $\Delta_{T'} = \|\hat{Q}^{\pi^*_{T'}} - Q^{\pi^*_{T'}}\|_\infty$. For any prior distribution $M_p$ on the transition model, any posterior $M_q$, any i.i.d. sampling distribution $\mathcal{U}$, with probability no less than $1 - \delta$ over the sampling of $U \sim \mathcal{U}$:*

$$\forall M_q : \mathbb{E}_{T' \sim M_q} \Delta_{T'} \quad \leq \quad \sqrt{\frac{D(M_q\|M_p) - \ln \delta + |S| \ln 2 + \ln |S| + \ln n_{\min}}{(n_{\min} - 1)k^2/2}}, \tag{6}$$

*where $n_{\min} = \min_{s,a} n_{s,a}$ and $D(.\|.)$ is the Kullback–Leibler (KL) divergence.*

The above theorem (proved in the Appendix) provides a lower bound on the expectation of the true value function when the policy is taken to be optimal according to the sampled model from the posterior:

$$\mathbb{E}Q^{\pi^*_{T'}} \geq \mathbb{E}\hat{Q}^{\pi^*_{T'}} - \tilde{O}\left(\sqrt{D(M_q\|M_p)/n_{\min}}\right). \tag{7}$$

This lower bound suggests a stochastic model-selection method in which one searches in the space of posteriors to maximize the bound. Notice that there are two elements to the above bound. One is the PAC part of the bound that suggests the selection of models with high empirical value functions for their optimal policy. There is also a penalty term (or a regularization term) that penalizes distributions that are far from the prior (the Bayesian side of the bound).

**Margin for Deterministic Policies**

One could apply Theorem 2 with any choice $M_q$. Generally, this will result in a bound on the value of a stochastic policy. However, if the optimal policy is the same for all of the possible samples from the posterior, then we will get a bound for that particular deterministic policy.

We define the *support of policy* $\pi$, denoted by $\mathcal{T}_\pi$, to be the set of transition models for which the optimal policy is $\pi$. Putting all the posterior probability on $\mathcal{T}_\pi$ will result in a tighter bound for the value of the policy $\pi$. The tightest bound occurs when $M_q$ is a scaled version of $M_p$ summing to 1 over $\mathcal{T}_\pi$, that is when we have:

$$M_q(T') = \begin{cases} \frac{M_p(T')}{M_p(\mathcal{T}_\pi)} & T' \in \mathcal{T}_\pi \\ 0 & T' \notin \mathcal{T}_\pi \end{cases} \tag{8}$$

In that case, the KL divergence is $D(M_q \| M_p) = -\ln M_p(\mathcal{T}_\pi)$, and the bound will be:

$$\mathbb{E} Q^{\pi^*_{T'}} \geq \mathbb{E}\hat{Q}^{\pi^*_{T'}} - \tilde{O}\left(\sqrt{-\ln M_p(\mathcal{T}_\pi)/n_{\min}}\right). \tag{9}$$

Intuitively, we will get tighter bounds for policies that have larger empirical values and higher prior probabilities supporting them.

Finding $M_p(\mathcal{T}_\pi)$ might not be computationally tractable. Therefore, we define a notion of *margin* for transition functions and policies and use it to get tractable bounds. The margin of a transition function $T'$, denoted by $\theta_{T'}$, is the maximum distance we can move away from $T'$ such that the optimal policy does not change:

$$\|T'' - T'\|_1 \leq \theta_{T'} \Rightarrow \pi^*_{T''} = \pi^*_{T'}. \tag{10}$$

The margin defines a hypercube around $T'$ for which the optimal policy does not change. In cases where the support set of a policy is difficult to find, one can use this hypercube to get a reasonable bound for the true value function of the corresponding policy. In that case, we would define the posterior to be the scaled prior defined only on the margin hypercube. The idea behind this method is similar to that of the Luckiness framework [11] and large-margin classifiers [16, 13]. This shows that the idea of maximizing margins can be applied to control problems as well as classification and regression tasks.

To find the margin of any given $T'$, if we know the value of the second best policy, we can calculate its regret according to $T'$ (it will be the smallest regret $\eta_{\min}$). Using Lemma 1, we can conclude that if $\|T'' - T'\|_1 \leq k\eta_{\min}/2$, then the value of the best and second best policies can change by at most $\eta_{\min}/2$, and thus the optimal policy will not change. Therefore, $\theta_{T'} \geq k\eta_{\min}/2$. One can then define the posterior on the transitions inside the margin to get a bound for the value function.

## 4 Model-Free PAC-Bayes Bound

In this section we introduce a PAC-Bayesian bound for model-free reinforcement learning on discrete state spaces. This time we assume that we are given a prior distribution on the space of value functions, rather than on transition models. This prior encodes an initial belief about the optimal value function for a given RL domain. This could be useful, for example, in the context of transfer learning, where one has learned a value function in one environment and then uses that as the prior belief on a similar domain.

We start by defining the TD error of a given value function $Q$ to be $\|Q - BQ\|_\infty$. In most cases, we do not have access to the Bellman optimality operator. When we only have access to a sample set $U$ collected on the RL domain, we can define the empirical Bellman optimality operator $\hat{B}$ to be:

$$\hat{B}Q(s,a) = \frac{1}{n_{s,a}} \sum_{(s,a,s',r) \in U} \left(r + \gamma \max_{a'} Q(s',a')\right), \tag{11}$$

Note that $\mathbb{E}[\hat{B}Q] = BQ$. We further make an assumption that all the $BQ$ values we could observe are bounded in the range $[c_{\min}, c_{\max}]$, with $c = c_{\max} - c_{\min}$. Using this assumption, one can use Hoeffding's inequality to bound the difference between the empirical and true Bellman operators:

$$\Pr\{|\hat{B}Q(s,a) - BQ(s,a)| > \epsilon\} \leq e^{-2n_{s,a}\epsilon^2/c^2}. \tag{12}$$

When the true Bellman operator is not known, one makes use of the empirical TD error, similarly defined to be $\|Q - \hat{B}Q\|_\infty$. Q-learning [14] and its derivations with function approximation [17], and also batch methods such as LSTD [18], often aim to minimize the empirical (projected) TD error. We argue that it might be better to choose a function that is not a fixed point of the empirical Bellman operator. Instead, we aim to minimize the upper bound on the approximation error (which might be referred to as loss) of the $Q$ function, as compared to the true optimal value.

The following theorem (proved in the Appendix) is the first PAC-Bayesian bound for model-free batch RL on discrete state spaces:

**Theorem 3.** *Let* $\Delta_Q = \|Q - Q^*\|_\infty - \frac{\|Q - \hat{B}Q\|_\infty}{1-\gamma}$. *For all prior distributions* $J_p$ *and posteriors* $J_q$ *over the space of value functions, with probability no less than* $1 - \delta$ *over the sampling of* $U \sim \mathcal{U}$:

$$\forall J_q : \mathbb{E}_{Q \sim J_q} \Delta_Q \leq \sqrt{\frac{D(J_q \| J_p) - \ln \delta + \ln |S| + \ln |A| + \ln n_{\min}}{2(n_{\min} - 1)(1 - \gamma)^2 / c^2}}. \tag{13}$$

This time we have an upper bound on the expected approximation error:

$$\mathbb{E}\|Q - Q^*\|_\infty \leq \frac{\mathbb{E}\|Q - \hat{B}Q\|_\infty}{1 - \gamma} + \tilde{O}\left(\sqrt{D(J_q \| J_p)/n_{\min}}\right). \tag{14}$$

This suggests a model-selection method in which one would search for a posterior $J_q$ to minimize the above bound. The PAC side of the bound guides this model-selection method to look for posteriors with smaller empirical TD error. The Bayesian part, on the other hand, penalizes the selection of posteriors that are far from the prior distribution.

One can use general forms of priors that would impose smoothness or sparsity for this model-selection technique. In that sense, this method would act as a regularization technique that penalizes complex and irregular functions. The idea of regularization in RL with function approximation is not new to this work [19]. This bound, however, is more general, as it could incorporate not only smoothness constraints, but also other forms of prior knowledge into the learning process.

## 5    Empirical Results

To illustrate the model-selection techniques based on the bounds in the paper, we consider one model-based RL domain and one model-free problem. The model-based domain is a chain model in which states are ordered by their index. The last state has a reward of $1$ and all other states have reward $0$. There are two types of actions. One is a stochastic "forward" operation which moves us to the next state in the chain with probability $0.5$ and otherwise makes a random transition. The second type is a stochastic "reset" which moves the system to the first state in the chain with probability $0.5$ and makes a random transition otherwise. In this domain, we have at each state two actions that do stochastic reset and one action that is a stochastic forward. There are $10$ states and $\gamma = 0.9$.

When there are only a few number of sample transitions for each state–action pair, there is a high chance that the frequentist estimate confuses a reset action with a forward. Therefore, we expect a good model-based prior to be useful in this case. We use independent Dirichlets as our prior. We experiment with priors for which the Dirichlet $\alpha$-vector sums up to $10$. We define our good prior to have $\alpha$-vectors proportional to the true transition probabilities. A misleading prior is one for which the vector is proportional to a transition model when the actions are switched between forward and reset. A weighted sum between the good and bad priors creates a range of priors that gradually change from being informative to misleading.

We compare the expected regret of three different methods. The empirical method uses the optimal policy with respect to the empirical models. The Bayesian method samples a transition model from the Bayesian Dirichlet posteriors (when the observed counts are added to the prior $\alpha$-vectors) and then uses the optimal policy with respect to the sampled model. The PAC-Bayesian method uses $counts + \lambda \alpha_{\text{prior}}$ as the $\alpha$-vector of the posterior and finds the value of $\lambda \in [0, 1]$, using linear search within values with distance $0.1$, that maximizes the lower bound of Theorem 2 (with a more optimistic value for $k$ and $\delta = 0.05$). It then samples from that distribution and uses the optimal policy with respect to the sampled model. The running time for a single run is a few seconds.

Figure 1 (left) shows the comparison between the maximum regret in these methods for different sample sizes when the prior is informative. This is averaged over 50 runs for the Bayesian and PAC-Bayesian methods and 10000 runs for the empirical method. The number of sampled transitions is the same for all state–action pairs. As expected, the Bayesian method outperforms the empirical one for small sample sizes. We can see that the PAC-Bayesian method is closely following the Bayesian one in this case. With a misleading prior, however, as we can see in Figure 1 (center), the empirical method outperforms the Bayesian one. This time, the regret rate of the PAC-Bayesian method follows that of the empirical method. Figure 1 (right) shows how the PAC-Bayesian method switches between following the empirical estimate and the Bayesian posterior as the prior gradually changes from being misleading to informative (four sample transitions per state action pair). This shows that the bound of Theorem 2 is helpful as a model selection technique.

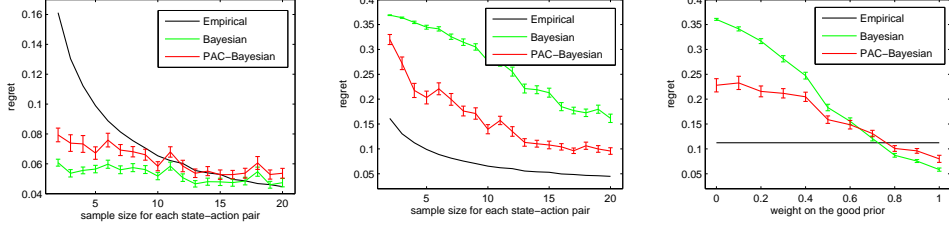

Figure 1: Average regrets of different methods. Error bars are 1 standard deviation of the mean.

The next experiment is to test the model-free bound of Theorem 3. The domain is a "puddle world". An agent moves around in a grid world of size $5 \times 9$ containing puddles with reward $-1$, an absorbing goal state with reward $+1$, and reward $0$ for the remaining states. There are stochastic actions along each of the four cardinal directions that move in the correct direction with probability $0.7$ and move in a random direction otherwise. If the agent moves towards the boundary then it stays in its current position.

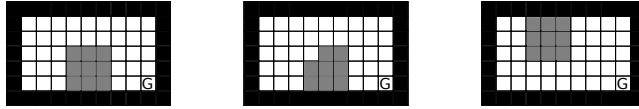

Figure 2: Maps of puddle world RL domain. Shaded boxes are puddles.

We first learn the true value function of a known prior map of the world (Figure 2, left). We then use that value function as the prior for our model-selection technique on two other environments. One of them is a similar environment where the shape of the puddle is slightly changed (Figure 2, center). We expect the prior to be informative and useful in this case. The other environment is, however, largely different from the first map (Figure 2, right). We thus expect the prior to be misleading.

Table 1: Performance of different model-selection methods.

|  | Empirical Regret | Bayesian Regret | PAC-Bayesian Regret | Average $\lambda$ |
|---|---|---|---|---|
| Similar Map | $0.21 \pm 0.03$ | $0.10 \pm 0.01$ | $0.12 \pm 0.01$ | $0.58 \pm 0.01$ |
| Different Map | $0.19 \pm 0.03$ | $1.16 \pm 0.09$ | $0.22 \pm 0.04$ | $0.03 \pm 0.03$ |

We start with independent Gaussians (one for each state–action pair) as the prior, with the initial map's $Q$-values for the mean $\mu_0$, and $\sigma_0^2 = 0.01$ for the variance. The posterior is chosen to be the product of Gaussians with mean $\left( \frac{\lambda \mu_0}{\sigma_0^2} + \frac{n \hat{Q}(.,.)}{\hat{\sigma}^2} \right) \Big/ \left( \frac{\lambda}{\sigma_0^2} + \frac{n}{\hat{\sigma}^2} \right)$ and variance $\left( \frac{\lambda}{\sigma_0^2} + \frac{n}{\hat{\sigma}^2} \right)^{-1}$, where $\hat{\sigma}^2$ is the empirical variance. We sample from this posterior and act according to its greedy policy. For $\lambda = 1$, this is the Bayesian posterior for the mean of a Gaussian with known variance. For $\lambda = 0$, the prior is completely ignored. We will, however, find the $\lambda \in [0, 1]$ that minimizes the PAC-Bayesian bound of Theorem 3 (with an optimistic choice of $c$ and $\delta = 0.05$) and compare it with the performance of the empirical policy and a semi-Bayesian policy that acts according to a sampled value from the Bayesian posterior.

Table 1 shows the average over 100 runs of the maximum regret for these methods and the average of the selected $\lambda$, with equal sample size of 20 per state–action pair. Again, it can be seen that the PAC-Bayesian method makes use of the prior (with higher values of $\lambda$) when the prior is informative, and otherwise follows the empirical estimate (smaller values of $\lambda$). It adaptively balances the usage of the prior based on its consistency over the observed data.

## 6 Discussion

This paper introduces the first set of PAC-Bayesian bounds for the batch RL problem in finite state spaces. We demonstrate how such bounds can be used for both model-based and model-free RL methods. Our empirical results show that PAC-Bayesian model-selection uses prior distributions when they are informative and useful, and ignores them when they are misleading.

For the model-based bound, we expect the running time of searching in the space of parametrized posteriors to increase rapidly with the size of the state space. A more scalable version would sample models around the posteriors, solve each model, and then use importance sampling to estimate the value of the bound for each possible posterior. This problem does not exist with the model-free approach, as we do not need to solve the MDP for each sampled model.

A natural extension to this work would be on domains with continuous state spaces, where one would use different forms of function approximation for the value function. There is also the possibility of future work in applications of PAC-Bayesian theorems in online reinforcement learning, where one targets the exploration–exploitation problem. Online PAC RL with Bayesian priors has recently been addressed with the BOSS algorithm [20]. PAC-Bayesian bounds could help derive similar model-free algorithms with theoretical guarantees.

**Acknowledgements:** Funding for this work was provided by the National Institutes of Health (grant R21 DA019800) and the NSERC Discovery Grant program.

## Appendix

The following lemma, due to McAllester [21], forms the basis of the proofs for both bounds:

**Lemma 4.** *For $\beta > 0$, $K > 0$, and $\mathcal{Q}, \mathcal{P}, \Delta \in R^n$ satisfying $\mathcal{P}_i, \mathcal{Q}_i, \Delta_i \geq 0$ and $\sum_{i=1}^n \mathcal{Q}_i = 1$:*

$$\sum_{i=1}^n \mathcal{P}_i e^{\beta \Delta_i^2} \leq K \qquad \Rightarrow \qquad \sum_{i=1}^n \mathcal{Q}_i \Delta_i \leq \sqrt{(D(\mathcal{Q}\|\mathcal{P}) + \ln K)/\beta}. \tag{15}$$

Note that even when we have arbitrary probability measures $\mathcal{Q}$ and $\mathcal{P}$ on a continuous space of $\Delta$'s, it might still be possible to define a sequence of vectors $\mathcal{Q}^{(1)}, \mathcal{Q}^{(2)}, \ldots, \mathcal{P}^{(1)}, \mathcal{P}^{(2)}, \ldots$ and $\Delta^{(1)}, \Delta^{(2)}, \ldots$ such that $\mathcal{Q}^{(n)}, \mathcal{P}^{(n)}$ and $\Delta^{(n)}$ satisfy the condition of the lemma and

$$\mathbb{E}_{\mathcal{Q}} \Delta = \lim_{n \to \infty} \sum_{i=1}^n \mathcal{Q}_i^{(n)} \Delta_i^{(n)}, \qquad D(\mathcal{Q}\|\mathcal{P}) = \lim_{n \to \infty} \sum_{i=1}^n \mathcal{Q}_i^{(n)} \ln \frac{\mathcal{Q}_i^{(n)}}{\mathcal{P}_i^{(n)}}. \tag{16}$$

We will then take the limit of the conclusion of the lemma to get a bound for the continuous case [21].

**Proof of Theorem 2 (Model-Based Bound)**

**Lemma 5.** *Let $\Delta_{T'} = \|\hat{Q}^{\pi_{T'}^*} - Q^{\pi_{T'}^*}\|_\infty$. With probability no less than $1 - \delta$ over the sampling:*

$$\mathbb{E}_{T' \sim M_p}[e^{\frac{1}{2}(n_{\min}-1)k^2 \Delta_{T'}^2}] \leq \frac{|S| 2^{|S|} n_{\min}}{\delta}. \tag{17}$$

Before proving Lemma 5, note that Lemma 5 and Lemma 4 together imply Therorem 2. We only need to apply the method described for arbitrary probability measures. To prove Lemma 5, it suffices to prove the following, swap the expectations and apply Markov's inequality:

$$\mathbb{E}_{T' \sim M_P} \mathbb{E}_{U \sim \mathcal{U}}[e^{\frac{1}{2}(n_{\min}-1)k^2 \Delta_{T'}^2}] \leq |S| 2^{|S|} n_{\min}. \tag{18}$$

Therefore, we only need to show that for any choice of $T'$, $\mathbb{E}_{U \sim \mathcal{U}}[e^{\frac{1}{2}(n_{\min}-1)k^2 \Delta_{T'}^2}]$ follows the bound. Let $a_s = \pi_{T'}^*(s)$. We have:

$$\Pr\{\Delta_{T'} \geq \epsilon\} \quad \leq \quad \sum_s \Pr\{\|\hat{T}(s, a_s, .) - T(s, a_s, .)\|_1 > k\epsilon\} \tag{19}$$

$$\leq \quad \sum_s \left(2^{|S|} e^{-\frac{1}{2} n_{s,a_s}(k\epsilon)^2}\right) \leq |S| 2^{|S|} e^{-\frac{1}{2} n_{\min}(k\epsilon)^2}. \tag{20}$$

The first line is by Lemma 1. The second line is a concentration inequality for multinomials [22]. We choose to maximize $\mathbb{E}_{U\sim\mathcal{U}}[e^{\frac{1}{2}(n_{\min}-1)k^2\Delta_{T'}^2}]$, subject to $\Pr\{\Delta_{T'}\geq\epsilon\}\leq|S|2^{|S|}e^{-\frac{1}{2}n_{\min}(k\epsilon)^2}$. The maximum occurs when the inequality is tight and the p.d.f. for $\Delta_{T'}$ is:

$$f(\Delta)=|S|2^{|S|}k^2 n_{\min}\Delta e^{-\frac{1}{2}n_{\min}k^2\Delta^2}. \tag{21}$$

We thus get:

$$\mathbb{E}_{U\sim\mathcal{U}}[e^{\frac{1}{2}(n_{\min}-1)k^2\Delta_{T'}^2}] \quad\leq\quad \int_0^\infty e^{\frac{1}{2}(n_{\min}-1)k^2\Delta^2}f(\Delta)d\Delta \tag{22}$$

$$= \int_0^\infty |S|2^{|S|}k^2 n_{\min}\Delta e^{-\frac{1}{2}k^2\Delta^2}d\Delta \leq |S|2^{|S|}n_{\min}. \tag{23}$$

This concludes the proof of Lemma 5 and consequently Theorem 2.

**Proof of Theorem 3 (Model-Free Bound)**

Since $B$ is a contraction with respect to the infinity norm and $Q^*$ is its fixed point, we have:

$$\|Q-Q^*\|_\infty = \|Q-BQ+BQ-BQ^*\|_\infty \leq \|Q-BQ\|_\infty+\|BQ-BQ^*\|_\infty \tag{24}$$

$$\leq \|Q-BQ\|_\infty+\gamma\|Q-Q^*\|_\infty \tag{25}$$

And thus $\|Q-Q^*\|_\infty\leq\frac{1}{1-\gamma}\|Q-BQ\|_\infty$.

**Lemma 6.** *Let* $\Delta_Q=\max(0,\|Q-Q^*\|_\infty-\frac{\|Q-\hat{B}Q\|_\infty}{1-\gamma})$. *With probability no less than* $1-\delta$:

$$\mathbb{E}_{Q\sim J_p}[e^{2(n_{\min}-1)(1-\gamma)^2\Delta_Q^2/c^2}]\leq\frac{|S||A|n_{\min}}{\delta}. \tag{26}$$

Similar to the previous section, Lemma 6 and Lemma 4 together imply Theorem 3.

To prove Lemma 6, similar to the previous proof, we only need to show that for any choice of $Q$, $\mathbb{E}_{U\sim\mathcal{U}}[e^{2(n_{\min}-1)(1-\gamma)^2\Delta_Q^2/c^2}]$ follows the bound. We have that:

$$\Pr\{\Delta_Q\geq\epsilon\}=\Pr\left\{\|Q-Q^*\|_\infty\geq\epsilon+\|Q-\hat{B}Q\|_\infty/(1-\gamma)\right\} \tag{27}$$

$$\leq \Pr\left\{\|Q-BQ\|_\infty\geq(1-\gamma)\left(\epsilon+\|Q-\hat{B}Q\|_\infty/(1-\gamma)\right)\right\} \tag{28}$$

$$\leq \sum_{s,a}\Pr\left\{|Q(s,a)-BQ(s,a)|\geq(1-\gamma)\epsilon+\|Q-\hat{B}Q\|_\infty\right\} \tag{29}$$

$$\leq \sum_{s,a}\Pr\left\{|Q(s,a)-\hat{B}Q(s,a)|+|\hat{B}Q(s,a)-BQ(s,a)|\geq(1-\gamma)\epsilon+\|Q-\hat{B}Q\|_\infty\right\} \tag{30}$$

$$\leq \sum_{s,a}\Pr\left\{|\hat{B}Q(s,a)-BQ(s,a)|\geq(1-\gamma)\epsilon\right\} \tag{31}$$

$$\leq \sum_{s,a}e^{-2n_{s,a}(1-\gamma)^2\epsilon^2/c^2}\leq|S||A|e^{-2n_{\min}(1-\gamma)^2\epsilon^2/c^2} \tag{32}$$

Eqn (28) follows from the derivations at the beginning of this section. Eqn (29) is by the union bound. Eqn (31) is by the definition of infinity norm. Last derivation is by Hoeffding inequality of Equation (12). Now again, similar to the model-based case, when the inequality is tight the p.d.f. is:

$$f(\Delta)=4|S||A|n_{\min}(1-\gamma)^2c^{-2}\Delta e^{-2n_{\min}(1-\gamma)^2\Delta^2/c^2}.$$

We thus get:

$$\mathbb{E}_{U\sim\mathcal{U}}[e^{2(n_{\min}-1)(1-\gamma)^2\Delta_Q^2/c^2}] \quad\leq\quad \int_0^\infty e^{2(n_{\min}-1)(1-\gamma)^2\Delta^2/c^2}f(\Delta)d\Delta$$

$$= \int_0^\infty 4|S||A|n_{\min}(1-\gamma)^2c^{-2}\Delta e^{-2(1-\gamma)^2\Delta^2/c^2}d\Delta$$

$$\leq |S||A|n_{\min}.$$

This concludes the proof of Lemma 6 and consequently Theorem 3.

## Footnotes

[1]This is a more general form of stochastic policy than is usually seen in the RL literature. A complete policy is sampled from an imposed distribution, correlating the selection of actions on different states.

# References

[1] L. G. Valiant. A theory of the learnable. *Commun. ACM*, 27(11):1134–1142, 1984.

[2] M. Kearns and S. Singh. Near-optimal reinforcement learning in polynomial time. *Machine Learning*, 49(2-3):209–232, 2002.

[3] R. I. Brafman and M. Tennenholtz. R-max – A general polynomial time algorithm for near-optimal reinforcement learning. *The Journal of Machine Learning Research*, 3:213–231, 2003.

[4] A. L. Strehl and M. L. Littman. A theoretical analysis of model-based interval estimation. In *Proceedings of the 22nd International Conference on Machine Learning*, pages 856–863, 2005.

[5] S. M. Kakade. *On the sample complexity of reinforcement learning*. PhD thesis, University College London, 2003.

[6] M. O. G. Duff. *Optimal learning: Computational procedures for Bayes-adaptive Markov decision processes*. PhD thesis, University of Massachusetts Amherst, 2002.

[7] M. J. A. Strens. A Bayesian Framework for Reinforcement Learning. In *Proceedings of the 17th International Conference on Machine Learning*, pages 943–950, 2000.

[8] T. Wang, D. Lizotte, M. Bowling, and D. Schuurmans. Bayesian sparse sampling for on-line reward optimization. In *Proceedings of the 22nd International Conference on Machine Learning*, page 963, 2005.

[9] J. Z. Kolter and A. Y. Ng. Near-Bayesian exploration in polynomial time. In *Proceedings of the 26th International Conference on Machine Learning*, pages 513–520, 2009.

[10] D. A. McAllester. Some PAC-Bayesian theorems. *Machine Learning*, 37(3):355–363, 1999.

[11] J. Shawe-Taylor and R. C. Williamson. A PAC analysis of a Bayesian estimator. In *Proceedings of the 10th Annual Conference on Computational Learning Theory*, pages 2–9, 1997.

[12] P. Germain, A. Lacasse, F. Laviolette, and M. Marchand. PAC-Bayesian learning of linear classifiers. In *Proceedings of the 26th International Conference on Machine Learning*, pages 353–360, 2009.

[13] J. Langford and J. Shawe-Taylor. PAC-Bayes and margins. In *Proceedings of Advances in Neural Information Processing Systems*, pages 439–446, 2002.

[14] R. S. Sutton and A. G. Barto. *Reinforcement Learning: An Introduction*. MIT Press, Cambridge, MA, 1998.

[15] D. P. Bertsekas and J. N. Tsitsiklis. *Neuro-Dynamic Programming (Optimization and Neural Computation Series, 3)*. Athena Scientific, 1996.

[16] R. Herbrich and T. Graepel. A PAC-Bayesian margin bound for linear classifiers. *IEEE Transactions on Information Theory*, 48(12):3140–3150, 2002.

[17] R. S. Sutton, D. McAllester, S. Singh, and Y. Mansour. Policy gradient methods for reinforcement learning with function approximation. *Proceedings of Advances in Neural Information Processing Systems*, 12:1057–1063, 2000.

[18] J. A. Boyan. Technical update: Least-squares temporal difference learning. *Machine Learning*, 49(2):233–246, 2002.

[19] A. Farahmand, M. Ghavamzadeh, C. Szepesvári, and S. Mannor. Regularized fitted Q-iteration: Application to planning. *Recent Advances in Reinforcement Learning*, pages 55–68, 2008.

[20] J. Asmuth, L. Li, M. L. Littman, A. Nouri, and D. Wingate. A Bayesian sampling approach to exploration in reinforcement learning. *The 25th Conference on Uncertainty in Artificial Intelligence*, 2009.

[21] D. A. McAllester. PAC-Bayesian model averaging. In *Proceedings of the 12th Annual Conference on Computational Learning Theory*, pages 164–170, 1999.

[22] T. Weissman, E. Ordentlich, G. Seroussi, S. Verdu, and M. J. Weinberger. Inequalities for the L1 deviation of the empirical distribution. Technical report, Information Theory Research Group, HP Laboratories, 2003.

